# Credit Assignment through Time: Alternatives to Backpropagation

**Yoshua Bengio** *
Dept. Informatique et
Recherche Opérationnelle
Université de Montréal
Montreal, Qc H3C-3J7

**Paolo Frasconi**
Dip. di Sistemi e Informatica
Universitá di Firenze
50139 Firenze (Italy)

## Abstract

Learning to recognize or predict sequences using long-term context has many applications. However, practical and theoretical problems are found in training recurrent neural networks to perform tasks in which input/output dependencies span long intervals. Starting from a mathematical analysis of the problem, we consider and compare alternative algorithms and architectures on tasks for which the span of the input/output dependencies can be controlled. Results on the new algorithms show performance qualitatively superior to that obtained with backpropagation.

## 1   Introduction

Recurrent neural networks have been considered to learn to map input sequences to output sequences. Machines that could efficiently learn such tasks would be useful for many applications involving sequence prediction, recognition or production.

However, practical difficulties have been reported in training recurrent neural networks to perform tasks in which the temporal contingencies present in the input/output sequences span long intervals. In fact, we can prove that dynamical systems such as recurrent neural networks will be increasingly difficult to train with gradient descent as the duration of the dependencies to be captured increases. A mathematical analysis of the problem shows that either one of two conditions arises in such systems. In the first case, the dynamics of the network allow it to reliably store bits of information (with bounded input noise), but gradients (with respect to an error at a given time step) vanish exponentially fast as one propagates them

backward in time. In the second case, the gradients can flow backward but the system is locally unstable and cannot reliably store bits of information in the presence of input noise.

In consideration of the above problem and the understanding brought by the theoretical analysis, we have explored and compared several alternative algorithms and architectures. Comparative experiments were performed on artificial tasks on which the span of the input/output dependencies can be controlled. In all cases, a duration parameter was varied, from $T/2$ to $T$, to avoid short sequences on which the algorithm could much more easily learn. These tasks require learning to *latch*, i.e. store bits of information for arbitrary durations (which may vary from example to example). Such tasks cannot be performed by Time Delay Neural Networks or by recurrent networks whose memories are gradually lost with time constants that are fixed by the parameters of the network.

Of all the alternatives to gradient descent that we have explored, an approach based on a probabilistic interpretation of a discrete state space, similar to hidden Markov models (HMMs), yielded the most interesting results.

## 2   A Difficult Problem of Error Propagation

Consider a non-autonomous discrete-time system with additive inputs, such as a recurrent neural network a with a continuous activation function:

$$a_t = M(a_{t-1}) + u_t \tag{1}$$

and the corresponding autonomous dynamics

$$a_t = M(a_{t-1}) \tag{2}$$

where $M$ is a nonlinear map (which may have tunable parameters such as network weights), and $a_t \in R^n$ and $u_t \in R^m$ are vectors representing respectively the system state and the external input at time $t$.

In order to latch a bit of state information one wants to restrict the values of the system activity $a_t$ to a subset $S$ of its domain. In this way, it will be possible to later interpret $a_t$ in at least two ways: inside $S$ and outside $S$. To make sure that $a_t$ remains in such a region, the system dynamics can be chosen such that this region is the basin of attraction of an attractor $X$ (or of an attractor in a sub-manifold or subspace of $a_t$'s domain). To "erase" that bit of information, the inputs may push the system activity $a_t$ out of this basin of attraction and possibly into another one.

In (Bengio, Simard, & Frasconi, 1994) we show that only two conditions can arise when using hyperbolic attractors to latch bits of information in such a system. Either the system is very sensitive to noise, or the derivatives of the cost at time $t$ with respect to the system activations $a_0$ converge exponentially to 0 as $t$ increases. This situation is the essential reason for the difficulty in using gradient descent to train a dynamical system to capture long-term dependencies in the input/output sequences.

A first theorem can be used to show that when the state $a_t$ is in a region where $|M'| > 1$, then small perturbations grow exponentially, which can yield to a loss of the information stored in the dynamics of the system:

**Theorem 1** *Assume $x$ is a point of $\mathbf{R}^n$ such that there exists an open sphere $U(x)$ centered on $x$ for which $|M'(z)| > 1$ for all $z \in U(x)$. Then there exist $y \in U(x)$ such that $\|M(x) - M(y)\| > \|x - y\|$.*

A second theorem shows that when the state $a_t$ is in a region where $|M'| < 1$, the gradients propagated backwards in time vanish exponentially fast:

**Theorem 2** *If the input $u_t$ is such that a system remains robustly latched ($|M'(a_t)| < 1$) on attractor $X$ after time 0, then $\frac{\partial a_t}{\partial a_0} \to 0$ as $t \to \infty$.*

See proofs in (Bengio, Simard, & Frasconi, 1994). A consequence of these results is that it is generally very difficult to train a parametric dynamical system (such as a recurrent neural network) to learn long-term dependencies using gradient descent. Based on the understanding brought by this analysis, we have explored and compared several alternative algorithms and architectures.

## 3   Global Search Methods

Global search methods such as simulated annealing can be applied to this problem, but they are generally *very* slow. We implemented the simulated annealing algorithm presented in (Corana, Marchesi, Martini, & Ridella, 1987) for optimizing functions of continuous variables. This is a "batch learning" algorithm (updating parameters after all examples of the training set have been seen). It performs a cycle of random moves, each along one coordinate (parameter) direction. Each point is accepted or rejected according to the Metropolis criterion (Kirkpatrick, Gelatt, & Vecchi, 1983). The simulated annealing algorithm is very robust with respect to local minima and long plateaus. Another global search method evaluated in our experiments is a multi-grid random search. The algorithm tries random points around the current solution (within a hyperrectangle of decreasing size) and accepts only those that reduce the error. Thus it is resistant to problems of plateaus but not as much resistant to problems of local minima. Indeed, we found the multi-grid random search to be much faster than simulated annealing but to fail on the parity problem, probably because of local minima.

## 4   Time Weighted Pseudo-Newton

The time-weighted pseudo-Newton algorithm uses second order derivatives of the cost with respect to each of the instantiations of a weight at different time steps to try correcting for the vanishing gradient problem. The weight update for a weight $w_i$ is computed as follows:

$$\Delta w_i(p) = -\sum_t \frac{\eta}{|\frac{\partial^2 C(p)}{\partial w_{it}^2}| + \mu} \times \frac{\partial C(p)}{\partial w_{it}} \tag{3}$$

where $w_{it}$ is the instantiation for time $t$ of parameter $w_i$, $\eta$ is a global learning rate and $C(p)$ is the cost for pattern $p$. In this way, each (temporal) contribution to $\Delta w_i(p)$ is weighted by the inverse curvature with respect to $w_{it}$. Like for the pseudo-Newton algorithm of Becker and Le Cun (1988) we prefer using a diagonal approximation of the Hessian which is cheap to compute and guaranteed to be positive definite.

The constant $\mu$ is introduced to prevent $\Delta w$ from becoming very large (when $|\frac{\partial^2 C(p)}{\partial w_{it}^2}|$ is very small). We found the performance of this algorithm to be better than the regular pseudo-Newton algorithm, which is better than the simple stochastic backpropagation algorithm, but all of these algorithms perform worse and worse as the length of the sequences is increased.

## 5   Discrete Error Propagation

The discrete error propagation algorithm replaces sigmoids in the network by discrete threshold units and attempts to propagate discrete error information backwards in time. The basic idea behind the algorithm is that for a simple discrete element such as a threshold unit or a latch, one can write down an error propagation rule that prescribes desired changes in the values of the inputs in order to obtain certain changes in the values of the outputs. In the case of a threshold unit, such a rule assumes that the desired change for the output of the unit is discrete (+2, 0 or -2). However, error information propagated backwards to such as unit might have a continuous value. A stochastic process is used to convert this continuous value into an appropriate discrete desired change. In the case of a self-loop, a clear advantage of this algorithm over gradient back-propagation through sigmoid units is that the error information does not vanish as it is repeatedly propagated backwards in time around the loop, even though the unit can robustly store a bit of information. Details of the algorithm will appear in (Bengio, Simard, & Frasconi, 1994). This algorithm performed better than the time-weighted pseudo-Newton, pseudo-Newton and back-propagation algorithms but the learning curve appeared very irregular, suggesting that the algorithm is doing a local random search.

## 6   An EM Approach to Target Propagation

The most promising of the algorithms we studied was derived from the idea of propagating targets instead of gradients. For this paper we restrict ourselves to sequence classification. We assume a finite-state learning system with the state $q_t$ at time $t$ taking on one of $n$ values. Different final states for each class are used as targets. The system is given a probabilistic interpretation and we assume a Markovian conditional independence model. As in HMMs, the system propagates forward a discrete distribution over the $n$ states. Transitions may be constrained so that each state $j$ has a defined set of successors $\mathcal{S}_j$.

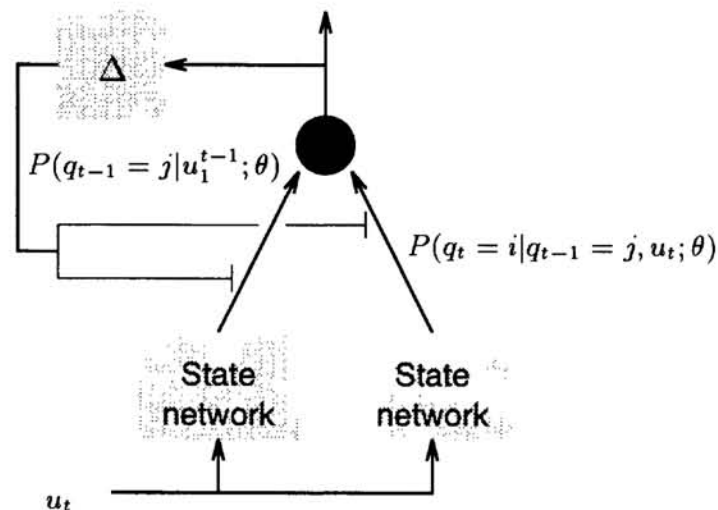

Figure 1: The proposed architecture

Learning is formulated as a maximum likelihood problem with missing data. Missing variables, over which an expectation is taken, are the paths in state-space. The

EM (Expectation/Maximization) or GEM (Generalized EM) algorithms (Dempster, Laird, & Rubin, 1977) can be used to help decoupling the influence of different hypothetical paths in state-space. The estimation step of EM requires propagating backward a discrete distribution of targets. In contrast to HMMs, where parameters are adjusted in an unsupervised learning framework, we use EM in a supervised fashion. This new perspective has been successful in training static models (Jordan & Jacobs, 1994).

Transition probabilities, conditional on the current input, can be computed by a parametric function such as a layer of a neural network with softmax units. We propose a modular architecture with one subnetwork $\mathcal{N}_j$ for each state (see Figure 1). Each subnetwork is feedforward, takes as input a continuous vector of features $u_t$ and has one output for each successor state, interpreted as $P(q_t = i \mid q_{t-1} = j, u_t; \theta)$, $(j = 1, \ldots, n, i \in \mathcal{S}_j)$. $\theta$ is a set of tunable parameters. Using a Markovian assumption, the distribution over states at time $t$ is thus obtained as a linear combination of the outputs of the subnetworks, gated by the previously computed distribution:

$$P(q_t = i \mid u_1^t; \theta) = \sum_j P(q_{t-1} = j \mid u_1^{t-1}; \theta) P(q_t = i \mid q_{t-1} = j, u_t; \theta) \qquad (4)$$

where $u_1^t$ is a subsequence of inputs from time 1 to $t$ inclusively. The training algorithm looks for parameters $\theta$ of the system that maximize the likelihood $L$ of falling in the "correct" state at the end of each sequence:

$$L(\theta) = \prod_p P(q_{T_p} = q_{T_p}^* \mid u_1^{T_p}; \theta) \qquad (5)$$

where $p$ ranges over training sequences, $T_p$ the length of the $p^{\text{th}}$ training sequence, and $q_{T_p}^*$ the desired state at time $T_p$.

An auxiliary function $Q(\theta, \theta^k)$ is constructed by introducing as hidden variables the whole state sequence, hence the complete likelihood function is defined as follows:

$$L_c(\theta) = \prod_p P(q_1^{T_p} \mid u_1^{T_p}; \theta) \qquad (6)$$

and

$$Q(\theta, \theta^k) = E[\log L_c(\theta) \mid \theta^k] \qquad (7)$$

where at the $k+1^{\text{th}}$ EM (or GEM) iteration, $\theta^{k+1}$ is chosen to maximize (or increase) the auxiliary function $Q$ with respect to $\theta$.

If the inputs are quantized and the subnetworks perform a simple look-up in a table of probabilities, then the EM algorithm can be used, i.e., $\frac{\partial Q(\theta, \theta^k)}{\partial \theta} = 0$ can be solved analytically. If the networks have non-linearities, (e.g., with hidden units and a softmax at their output to constrain the outputs to sum to 1), then one can use the GEM algorithm (which simply increases $Q$, for example with gradient ascent) or directly perform (preferably stochastic) gradient ascent on the likelihood.

An extra term was introduced in the optimization criterion when we found that in many cases the target information would not propagate backwards (or would be diffused over all the states). These experiments confirmed previous results indicating a general difficulty of training fully connected HMMs, with the EM algorithm converging very often to poor local maxima of the likelihood. In an attempt to understand better the phenomenon, we looked at the quantities propagated forward and the quantities propagated backward (representing credit or blame) in the

training algorithm. We found a diffusion of credit or blame occurring when the forward maps (i.e. the matrix of transition probabilities) at each time step are such that many inputs map to a few outputs, i.e., when the ratio of a small volume in the image of the map with respect to the corresponding volume in the domain is small. This ratio is the absolute value of the determinant of the Jacobian of the map. Hence, using an optimization criterion that incorporates the maximization of the average magnitude of the determinant of the transition matrices, this algorithm performs much better than the other algorithms. Two other tricks were found to be important to help convergence and reduce the problem of diffusion of credit. The first idea is to use whenever possible a structured model with a sparse connectivity matrix, thus introducing some prior knowledge about the state-space. For example, applications of HMMs to speech recognition always rely on such structured topologies. We could reduce connectivity in the transition matrix for the 2-sequence problem (see next section for its definition) by splitting some of the nodes into two subsets, each specializing on one of the sequence classes. However, sometimes it is not possible to introduce such constraints, such as in the parity problem. Another trick that drastically improved performance was to use stochastic gradient ascent in a way that helps the training algorithm get out of local optima. The learning rate is decreased when the likelihood improves but it is increased when the likelihood remains flat (the system is stuck in a plateau or local optimum).

As the results in the next section show, the performances obtained with this algorithm are much better than those obtained with the other algorithms on the two simple test problems that were considered.

## 7    Experimental Results

We present here results on two problems for which one can control the span of input/output dependencies. The 2-sequence problem is the following: classify an input sequence, at the end of the sequence, in one of two types, when only the first $N$ elements ($N = 3$ in our experiments) of this sequence carry information about the sequence class. Uniform noise is added to the sequence. For the first 6 methods (see Tables 1 to 4), we used a fully connected recurrent network with 5 units (with 25 free parameters). For the EM algorithm, we used a 7-state system with a sparse connectivity matrix (an initial state, and two separate left-to-right submodels of three states each to model the two types of sequences).

The parity problem consists in producing the parity of an input sequence of 1's and -1's (i.e., a 1 should be produced at the final output if and only if the number of 1's in the input is odd). The target is only given at the end of the sequence. For the first 6 methods we used a minimal size network (1 input, 1 hidden, 1 output, 7 free parameters). For the EM algorithm, we used a 2-state system with a full connectivity matrix.

Initial parameters were chosen randomly for each trial. Noise added to the sequence was also uniformly distributed and chosen independently for each training sequence. We considered two criteria: (1) the average classification error at the end of training, i.e., after a stopping criterion has been met (when either some allowed number of function evaluations has been performed or the task has been learned), (2) the average number of function evaluations needed to reach the stopping criterion.

In the tables, "p-n" stands for pseudo-Newton. Each column corresponds to a value of the maximum sequence length $T$ for a given set of trials. The sequence length for a particular training sequence was picked randomly within $T/2$ and $T$. Numbers

reported are averages over 20 or more trials.

# 8   Conclusion

Recurrent networks and other parametric dynamical systems are very powerful in their ability to represent and use context. However, theoretical and experimental evidence shows the difficulty of assigning credit through many time steps, which is required in order to *learn* to use and represent context. This paper studies this fundamental problem and proposes alternatives to the backpropagation algorithm to perform such learning tasks. Experiments show these alternative approaches to perform significantly better than gradient descent. The behavior of these algorithms yields a better understanding of the central issue of learning to use context, or assigning credit through many transformations. Although all of the alternative algorithms presented here showed some improvement with respect to standard stochastic gradient descent, a clear winner in our comparison was an algorithm based on the EM algorithm and a probabilistic interpretation of the system dynamics. However, experiments on more challenging tasks will have to be conducted to confirm those results. Furthermore, several extensions of this model are possible, for example allowing both inputs and outputs, with supervision on outputs rather than on states. Finally, similarly to the work we performed for recurrent networks trained with gradient descent, it would be very important to analyze theoretically the problems of propagation of credit encountered in training such Markov models.

## Acknowledgements

We wish to emphatically thank Patrice Simard, who collaborated with us on the analysis of the theoretical difficulties in learning long-term dependencies, and on the discrete error propagation algorithm.

## Footnotes

*also, AT&T Bell Labs, Holmdel, NJ 07733

## References

S. Becker and Y. Le Cun. (1988) Improving the convergence of back-propagation learning with second order methods, *Proc. of the 1988 Connectionist Models Summer School*, (eds. Touretzky, Hinton and Sejnowski), Morgan Kaufman, pp. 29–37.

Y. Bengio, P. Simard, and P. Frasconi. (1994) Learning long-term dependencies with gradient descent is difficult, *IEEE Trans. Neural Networks*, (in press).

A. Corana, M. Marchesi, C. Martini, and S. Ridella. (1987) Minimizing multimodal functions of continuous variables with the simulated annealing algorithm, *ACM Transactions on Mathematical Software*, vol. 13, no. 13, pp. 262–280.

A.P. Dempster, N.M. Laird, and D.B. Rubin. (1977) Maximum-likelihood from incomplete data via the EM algorithm, *J. of Royal Stat. Soc.*, vol. B39, pp. 1–38.

M.I. Jordan and R.A. Jacobs. (1994) Hierarchical mixtures of experts and the EM algorithm, *Neural Computation*, (in press).

S. Kirkpatrick, C.D. Gelatt, and M.P. Vecchi. (1983) Optimization by simulated annealing, *Science 220*, 4598, pp.671–680.

Table 1: Final classification error for the 2-sequence problem wrt sequence length

|  | 5 | 10 | 20 | 50 | 100 |
|---|---|---|---|---|---|
| back-prop | 58 | 56 | 43 | 53 | 50 |
| p-n | 2 | 3 | 10 | 25 | 29 |
| time-weighted p-n | 0 | 0 | 9 | 34 | 14 |
| multigrid | 2 | 6 | 1 | 3 | 6 |
| discrete err. prop. | 6 | 16 | 29 | 23 | 22 |
| simulated anneal. | 6 | 0 | 7 | 4 | 11 |
| EM | 0 | 0 | 0 | 0 | 0 |

Table 2: # sequence presentations for the 2-sequence problem wrt sequence length

|  | 5 | 10 | 20 | 50 | 100 |
|---|---|---|---|---|---|
| back-prop | 2.9e3 | 3.0e3 | 2.9e3 | 3.0e3 | 2.8e3 |
| p-n | 5.1e2 | 1.1e3 | 1.9e3 | 2.6e3 | 2.5e3 |
| time-weighted p-n | 5.4e2 | 4.3e2 | 2.4e3 | 2.9e3 | 2.7e3 |
| multigrid | 4.1e3 | 5.8e3 | 2.5e3 | 3.9e3 | 6.4e3 |
| discrete err. prop. | 6.6e2 | 1.3e3 | 2.1e3 | 2.1e3 | 2.1e3 |
| simulated anneal. | 2.0e5 | 3.9e4 | 8.2e4 | 7.7e4 | 4.3e4 |
| EM | 3.2e3 | 4.0e3 | 2.9e3 | 3.2e3 | 2.9e3 |

Table 3: Final classification error for the parity problem wrt sequence length

|  | 3 | 5 | 10 | 20 | 50 | 100 | 500 |
|---|---|---|---|---|---|---|---|
| back-prop | 2 | 20 | 41 | 38 | 43 |  |  |
| p-n | 3 | 25 | 41 | 44 | 40 | 47 |  |
| time-weighted p-n | 26 |  | 39 | 43 | 44 |  |  |
| multigrid | 15 |  |  | 44 | 45 |  |  |
| discrete err. prop. | 0 |  | 0 | 0 | 5 |  |  |
| simulated anneal. | 3 |  |  | 10 | 0 |  |  |
| EM |  | 0 | 6 | 0 | 14 | 0 | 12 |

Table 4: # sequence presentations for the parity problem wrt sequence length

|  | 3 | 5 | 9 | 20 | 50 | 100 | 500 |
|---|---|---|---|---|---|---|---|
| back-prop | 3.6e3 | 5.5e3 | 8.7e3 | 1.6e4 | 1.1e4 |  |  |
| p-n | 2.5e2 | 8.9e3 | 8.9e3 | 7.7e4 | 1.1e4 | 1.1e5 |  |
| time-weighted p-n | 4.5e4 |  | 7.0e4 | 3.4e4 | 8.1e4 |  |  |
| multigrid | 4.2e3 |  |  | 1.5e4 | 3.1e4 |  |  |
| discrete err. prop. | 5.0e3 |  | 7.9e3 | 1.5e4 | 5.4e4 |  |  |
| simulated anneal. | 5.1e5 |  |  | 1.2e6 | 8.1e5 |  |  |
| EM |  | 2.3e3 | 1.5e3 | 1.3e3 | 3.2e3 | 2.6e3 | 3.4e3 |